# Bayesian Co-Training

**Shipeng Yu, Balaji Krishnapuram, Romer Rosales, Harald Steck, R. Bharat Rao**
CAD & Knowledge Solutions, Siemens Medical Solutions USA, Inc.
`firstname.lastname@siemens.com`

## Abstract

We propose a Bayesian undirected graphical model for co-training, or more generally for semi-supervised multi-view learning. This makes explicit the previously unstated assumptions of a large class of co-training type algorithms, and also clarifies the circumstances under which these assumptions fail. Building upon new insights from this model, we propose an improved method for co-training, which is a novel co-training kernel for Gaussian process classifiers. The resulting approach is convex and avoids local-maxima problems, unlike some previous multi-view learning methods. Furthermore, it can automatically estimate how much each view should be trusted, and thus accommodate noisy or unreliable views. Experiments on toy data and real world data sets illustrate the benefits of this approach.

## 1 Introduction

Data samples may sometimes be characterized in multiple ways, e.g., web-pages can be described both in terms of the textual content in each page and the hyperlink structure between them. [1] have shown that the error rate on unseen test samples can be upper bounded by the disagreement between the classification-decisions obtained from independent characterizations (i.e., *views*) of the data. Thus, in the web-page example, *misclassification rate* can be indirectly minimized by reducing the *rate of disagreement* between hyperlink-based and content-based classifiers, provided these characterizations are independent conditional on the class.

In many application domains class labels can be expensive to obtain and hence scarce, whereas unlabeled data are often cheap and abundantly available. Moreover, the disagreement between the class labels suggested by different views can be computed even when using unlabeled data. Therefore, a natural strategy for using unlabeled data to minimize the misclassification rate is to enforce consistency between the classification decisions based on several independent characterizations of the unlabeled samples. For brevity, unless otherwise specified, we shall use the term *co-training* to describe the entire genre of methods that rely upon this intuition, although strictly it should only refer to the original algorithm of [2].

Some co-training algorithms jointly optimize an objective function which includes misclassification penalties (loss terms) for classifiers from each view and a regularization term that penalizes lack of agreement between the classification decisions of the different views. In recent times, this *co-regularization* approach has become the dominant strategy for exploiting the intuition behind multi-view consensus learning, rendering obsolete earlier alternating-optimization strategies.

We survey in Section 2 the major approaches to co-training, the theoretical guarantees that have spurred interest in the topic, and the previously published concerns about the applicability to certain domains. We analyze the precise assumptions that have been made and the optimization criteria to better understand why these approaches succeed (or fail) in certain situations. Then in Section 3 we propose a principled undirected graphical model for co-training which we call the *Bayesian co-training*, and show that co-regularization algorithms provide one way for maximum-likelihood (ML) learning under this probabilistic model. By explicitly highlighting previously unstated assumptions,

Bayesian co-training provides a deeper understanding of the co-regularization framework, and we are also able to discuss certain fundamental limitations of multi-view consensus learning. In Section 4, we show that even simple and visually illustrated 2-D problems are sometimes not amenable to a co-training/co-regularization solution (no matter which specific model/algorithm is used – including ours). Empirical studies on two real world data sets are also illustrated.

Summarizing our algorithmic contributions, co-regularization is exactly equivalent to the use of a novel *co-training kernel* for *support vector machines* (SVMs) and *Gaussian processes* (GP), thus allowing one to leverage the large body of available literature for these algorithms. The kernel is intrinsically *non-stationary*, i.e., the level of similarity between any pair of samples depends on *all* the available samples, whether labeled or unlabeled, thus promoting semi-supervised learning. Therefore, this approach is significantly simpler and more efficient than the alternating-optimization that is used in previous co-regularization implementations. Furthermore, we can automatically estimate how much each view should be trusted, and thus accommodate noisy or unreliable views.

## 2 Related Work

**Co-Training and Theoretical Guarantees:** The iterative, alternating co-training method originally introduced in [2] works in a bootstrap mode, by repeatedly adding pseudo-labeled unlabeled samples into the pool of labeled samples, retraining the classifiers for each view, and pseudo-labeling additional unlabeled samples where at least one view is confident about its decision. The paper provided PAC-style guarantees that if (a) there exist weakly useful classifiers on each view of the data, and (b) these characterizations of the sample are conditionally independent given the class label, then the co-training algorithm can utilize the unlabeled data to learn arbitrarily strong classifiers.

[1] proved PAC-style guarantees that if (a) sample sizes are large, (b) the different views are conditionally independent given the class label, and (c) the classification decisions based on multiple views largely agree with each other, then with high probability the misclassification rate is upper bounded by the rate of disagreement between the classifiers based on each view. [3] tried to reduce the strong theoretical requirements. They showed that co-training would be useful if (a) there exist low error rate classifiers on each view, (b) these classifiers never make mistakes in classification when they are confident about their decisions, and (c) the two views are not too highly correlated, in the sense that there would be at least some cases where one view makes confident classification decisions while the classifier on the other view does not have much confidence in its own decision.

While each of these theoretical guarantees is intriguing and theoretically interesting, they are also rather unrealistic in many application domains. The assumption that classifiers do not make mistakes when they are confident and that of class conditional independence are rarely satisfied in practice. Nevertheless empirical success has been reported.

**Co-EM and Related Algorithms:** The Co-EM algorithm of [4] extended the original bootstrap approach of the co-training algorithm to operate simultaneously on all unlabeled samples in an iterative batch mode. [5] used this idea with SVMs as base classifiers and subsequently in unsupervised learning by [6]. However, co-EM also suffers from local maxima problems, and while each iteration's optimization step is clear, the co-EM is not really an expectation maximization algorithm (i.e., it lacks a clearly defined overall log-likelihood that monotonically improves across iterations).

**Co-Regularization:** [7] proposed an approach for two-view consensus learning based on simultaneously learning multiple classifiers by maximizing an objective function which penalized misclassifications by any individual classifier, and included a regularization term that penalized a high level of disagreement between different views. This co-regularization framework improves upon the co-training and co-EM algorithms by maximizing a convex objective function; however the algorithm still depends on an alternating optimization that optimizes one view at a time. This approach was later adapted to two-view spectral clustering [8].

**Relationship to Current Work:** The present work provides a probabilistic graphical model for multi-view consensus learning; alternating optimization based co-regularization is shown to be just one algorithm that accomplishes ML learning in this model. A more efficient, alternative strategy is proposed here for fully Bayesian classification under the same model. In practice, this strategy offers several advantages: it is easily extended to multiple views, it accommodates noisy views which are less predictive of class labels, and reduces run-time and memory requirements.

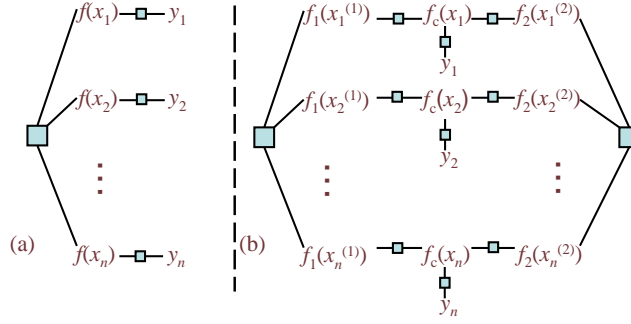

Figure 1: Factor graph for (a) one-view and (b) two-view models.

## 3 Bayesian Co-Training

### 3.1 Single-View Learning with Gaussian Processes

A Gaussian Process (GP) defines a nonparametric prior over functions in Bayesian statistics [9]. A random real-valued function $f : \mathbb{R}^d \to \mathbb{R}$ follows a GP, denoted by $\mathcal{GP}(h, \kappa)$, if for every finite number of data points $x_1, \ldots, x_n \in \mathbb{R}^d$, $\mathbf{f} = \{f(x_i)\}_{i=1}^n$ follows a multivariate Gaussian $\mathcal{N}(\mathbf{h}, \mathbf{K})$ with mean $\mathbf{h} = \{h(x_i)\}_{i=1}^n$ and covariance $\mathbf{K} = \{\kappa(x_i, x_j)\}_{i,j=1}^n$. Normally we fix the mean function $h \equiv 0$, and take a parametric (and usually stationary) form for the kernel function $\kappa$ (e.g., the Gaussian kernel $\kappa(x_k, x_\ell) = \exp(-\rho\|x_k - x_\ell\|^2)$ with $\rho > 0$ a free parameter).

In a single-view, supervised learning scenario, an output or target $y_i$ is given for each observation $x_i$ (e.g., for regression $y_i \in \mathbb{R}$ and for classification $y_i \in \{-1, +1\}$). In the GP model we assume there is a latent function $f$ underlying the output, $p(y_i|x_i) = \int p(y_i|f, x_i)p(f)\, df$, with the GP prior $p(f) = \mathcal{GP}(h, \kappa)$. Given the latent function $f$, $p(y_i|f, x_i) = p(y_i|f(x_i))$ takes a Gaussian noise model $\mathcal{N}(f(x_i), \sigma^2)$ for regression, and a sigmoid function $\lambda(y_i f(x_i))$ for classification.

The dependency structure of the single-view GP model can be shown as an undirected graph as in Fig. 1(a). The maximal cliques of the graphical model are the fully connected nodes $(f(x_1), \ldots, f(x_n))$ and the pairs $(y_i, f(x_i))$, $i = 1, \ldots, n$. Therefore, the joint probability of random variables $\mathbf{f} = \{f(x_i)\}$ and $\mathbf{y} = \{y_i\}$ is defined as $p(\mathbf{f}, \mathbf{y}) = \frac{1}{Z}\psi(\mathbf{f})\prod_i \psi(y_i, f(x_i))$, with potential functions[1]

$$\psi(\mathbf{f}) = \exp(-\tfrac{1}{2}\mathbf{f}^\top \mathbf{K}^{-1}\mathbf{f}), \quad \psi(y_i, f(x_i)) = \begin{cases} \exp(-\frac{1}{2\sigma^2}\|y_i - f(x_i)\|^2) & \text{for regression} \\ \lambda(y_i f(x_i)) & \text{for classification} \end{cases} \quad (1)$$

and normalization factor $Z$ (hereafter $Z$ is defined such that the joint probability sums to 1).

### 3.2 Undirected Graphical Model for Multi-View Learning

In multi-view learning, suppose we have $m$ different views of a same set of $n$ data samples. Let $x_i^{(j)} \in \mathbb{R}^{d_j}$ be the features for the $i$-th sample obtained using the $j$-th view, where $d_j$ is the dimensionality of the input space for view $j$. Then the vector $x_i \triangleq (x_i^{(1)}, \ldots, x_i^{(m)})$ is the complete representation of the $i$-th data sample, and $x^{(j)} \triangleq (x_1^{(j)}, \ldots, x_n^{(j)})$ represents all sample observations for the $j$-th view. As in the single-view learning, let $\mathbf{y} = (y_1, \ldots, y_n)$ where $y_i$ is the single output assigned to the $i$-th data point.

One can clearly concatenate the multiple views into a single view and apply a single-view GP model, but the basic idea of multi-view learning is to introduce *one function per view* which only uses the features from that view, and then jointly optimize these functions such that they come to a consensus. Looking at this problem from a GP perspective, let $f_j$ denote the latent function for the $j$-th view (i.e., using features only from view $j$), and let $f_j \sim \mathcal{GP}(0, \kappa_j)$ be its GP prior in view $j$. Since one data sample $i$ has only one single label $y_i$ even though it has multiple features from the multiple

views (i.e., latent function value $f_j(x_i^{(j)})$ for view $j$), the label $y_i$ should depend on *all* of these latent function values for data sample $i$.

The challenge here is to make this dependency explicit in a graphical model. We tackle this problem by introducing a new latent function, the *consensus function* $f_c$, to ensure conditional independence between the output $y$ and the $m$ latent functions $\{f_j\}$ for the $m$ views (see Fig. 1(b) for the undirected graphical model). At the functional level, the output $y$ depends *only* on $f_c$, and latent functions $\{f_j\}$ depend on each other *only via* the consensus function $f_c$. That is, we have the joint probability:

$$p(y, f_c, f_1, \ldots, f_m) = \frac{1}{Z} \psi(y, f_c) \prod_{j=1}^{m} \psi(f_j, f_c),$$

with some potential functions $\psi$. In the ground network with $n$ data samples, let $\mathbf{f}_c = \{f_c(x_i)\}_{i=1}^n$ and $\mathbf{f}_j = \{f_j(x_i^{(j)})\}_{i=1}^n$. The graphical model leads to the following factorization:

$$p(\mathbf{y}, \mathbf{f}_c, \mathbf{f}_1, \ldots, \mathbf{f}_m) = \frac{1}{Z} \prod_i \psi(y_i, f_c(x_i)) \prod_{j=1}^{m} \psi(\mathbf{f}_j) \psi(\mathbf{f}_j, \mathbf{f}_c). \tag{2}$$

Here the *within-view potential* $\psi(\mathbf{f}_j)$ specifies the dependency structure within each view $j$, and the *consensus potential* $\psi(\mathbf{f}_j, \mathbf{f}_c)$ describes how the latent function in each view is related with the consensus function $f_c$. With a GP prior for each of the views, we can define the following potentials:

$$\psi(\mathbf{f}_j) = \exp\left(-\frac{1}{2}\mathbf{f}_j^\top \mathbf{K}_j^{-1} \mathbf{f}_j\right), \quad \psi(\mathbf{f}_j, \mathbf{f}_c) = \exp\left(-\frac{\|\mathbf{f}_j - \mathbf{f}_c\|^2}{2\sigma_j^2}\right), \tag{3}$$

where $\mathbf{K}_j$ is the covariance matrix of view $j$, i.e., $\mathbf{K}_j(x_k, x_\ell) = \kappa_j(x_k^{(j)}, x_\ell^{(j)})$, and $\sigma_j > 0$ a scalar which quantifies how far away the latent function $\mathbf{f}_j$ is from $\mathbf{f}_c$. The *output potential* $\psi(y_i, f_c(x_i))$ is defined the same as that in (1) for regression or classification.

Some more insight may be gained by taking a careful look at these definitions: 1) The within-view potentials only rely on the *intrinsic structure* of each view, i.e., through the covariance $\mathbf{K}_j$ in a GP setting; 2) Each consensus potential actually defines a Gaussian over the difference of $\mathbf{f}_j$ and $\mathbf{f}_c$, i.e., $\mathbf{f}_j - \mathbf{f}_c \sim \mathcal{N}(\mathbf{0}, \sigma_j^2 \mathbf{I})$, and it can also be interpreted as assuming a conditional Gaussian for $\mathbf{f}_j$ with the consensus $\mathbf{f}_c$ being the mean. Alternatively if we focus on $\mathbf{f}_c$, the joint consensus potentials effectively define a conditional Gaussian prior for $\mathbf{f}_c$, $\mathbf{f}_c | \mathbf{f}_1, \ldots, \mathbf{f}_m$, as $\mathcal{N}(\boldsymbol{\mu}_c, \sigma_c^2 \mathbf{I})$ where

$$\boldsymbol{\mu}_c = \sigma_c^2 \sum_j \frac{\mathbf{f}_j}{\sigma_j^2}, \qquad \sigma_c^2 = \left(\sum_j \frac{1}{\sigma_j^2}\right)^{-1}. \tag{4}$$

This can be easily verified as a product of Gaussians. This indicates that the prior mean of the consensus function $\mathbf{f}_c$ is a *weighted combination* of the latent functions from all the views, and the weight is given by the inverse variance of each consensus potential. The higher the variance, the smaller the contribution to the consensus function.

More insights of this undirected graphical model can be seen from the marginals, which we discuss in detail in the following subsections. One advantage of this representation is that is allows us to see that many existing multi-view learning models are actually a special case of the proposed framework. In addition, this Bayesian interpretation also helps us understand both the benefits and the limitations of co-training.

### 3.3 Marginal 1: Co-Regularized Multi-View Learning

By taking the integral of (2) over $\mathbf{f}_c$ (and ignoring the output potential for the moment), we obtain the joint marginal distribution of the $m$ latent functions:

$$p(\mathbf{f}_1, \ldots, \mathbf{f}_m) = \frac{1}{Z} \exp\left\{-\frac{1}{2}\sum_{j=1}^{m} \mathbf{f}_j \mathbf{K}_j^{-1} \mathbf{f}_j - \frac{1}{2}\sum_{j<k} \frac{\|\mathbf{f}_j - \mathbf{f}_k\|^2}{\sigma_j^2 + \sigma_k^2}\right\}. \tag{5}$$

It is clearly seen that the negation of the logarithm of this marginal exactly recovers the regularization terms in *co-regularized multi-view learning*: The first part regularizes the functional space of each

view, and the second part constrains that all the functions need to agree on their outputs (inversely weighted by the sum of the corresponding variances).

From the GP perspective, (5) actually defines a *joint multi-view prior* for the $m$ latent functions, $(\mathbf{f}_1, \ldots, \mathbf{f}_m) \sim \mathcal{N}(\mathbf{0}, \mathbf{\Lambda}^{-1})$, where $\mathbf{\Lambda}$ is a $mn \times mn$ matrix with block-wise definition

$$\mathbf{\Lambda}(j,j) = \mathbf{K}_j^{-1} + \sum_{k \neq j} \frac{1}{\sigma_j^2 + \sigma_k^2}\mathbf{I}, \quad \mathbf{\Lambda}(j,j') = -\frac{1}{\sigma_j^2 + \sigma_{j'}^2}\mathbf{I}, \quad j = 1, \ldots, m, \quad j' \neq j. \quad (6)$$

Jointly with the target variable $\mathbf{y}$, the marginal is (for instance for regression):

$$p(\mathbf{y}, \mathbf{f}_1, \ldots, \mathbf{f}_m) = \frac{1}{Z} \exp \left\{ -\frac{1}{2}\sum_j \frac{\|\mathbf{f}_j - \mathbf{y}\|^2}{\sigma_j^2 + \sigma^2} - \frac{1}{2}\sum_{j=1}^m \mathbf{f}_j \mathbf{K}_j^{-1} \mathbf{f}_j - \frac{1}{2}\sum_{j<k} \frac{\|\mathbf{f}_j - \mathbf{f}_k\|^2}{\sigma_j^2 + \sigma_k^2} \right\}. \quad (7)$$

This recovers the co-regularization with least square loss in its log-marginal form.

### 3.4 Marginal 2: The Co-Training Kernel

The joint multi-view kernel defined in (6) is interesting, but it has a large dimension and is difficult to work with. A more interesting kernel can be obtained if we instead integrate out all the $m$ latent functions in (2). This leads to a Gaussian prior $p(\mathbf{f}_c) = \mathcal{N}(\mathbf{0}, \mathbf{K}_c)$ for the consensus function $f_c$, where

$$\mathbf{K}_c = \left[ \sum_j (\mathbf{K}_j + \sigma_j^2 \mathbf{I})^{-1} \right]^{-1}. \quad (8)$$

In the following we call $\mathbf{K}_c$ the *co-training kernel* for multi-view learning. This marginalization is very important, because it reveals the previously unclear insight of how the kernels from different views are combined together in a multi-view learning framework. This allows us to transform a multi-view learning problem into a single-view problem, and simply use the co-training kernel $\mathbf{K}_c$ to solve GP classification or regression. Since this marginalization is equivalent to (5), we will end up with solutions that are largely similar to any other co-regularization algorithm, but however a key difference is the Bayesian treatement contrasting previous ML-optimization methods. Additional benefits of the co-training kernel include the following:

1. The co-training kernel avoids repeated alternating optimizations over the different views $\mathbf{f}_j$, and directly works with a single consensus view $\mathbf{f}_c$. This reduces both time complexity and space complexity (only maintains $\mathbf{K}_c$ in memory) of multi-view learning.

2. While other alternating optimization algorithms might converge to local minima (because they optimize, not integrate), the single consensus view guarantees the *global optimal solution* for multi-view learning.

3. Even if all the individual kernels are stationary, $\mathbf{K}_c$ is in general *non-stationary*. This is because the inverse-covariances are added and then inverted again. In a transductive setting where the data are partially labeled, the co-training kernel between labeled data is also dependent on the unlabeled data. Hence the proposed co-training kernel can be used for semi-supervised GP learning [10].

### 3.5 Benefits of Bayesian Co-Training

The proposed undirected graphical model provides better understandings of multi-view learning algorithms. The co-training kernel in (8) indicates that *the Bayesian co-training is equivalent to single-view learning with a special (non-stationary) kernel*. This is also the preferable way of working with multi-view learning since it avoids alternating optimizations. Here are some other benefits which are not mentioned before:

**Trust-worthiness of each view:** The graphical model allows each view $j$ to have its own levels of uncertainty (or trust-worthiness) $\sigma_j^2$. In particular, a larger value of $\sigma_j^2$ implies less confidence on the observation of evidence provided by the $j$-th view. Thus when some views of the data are better at predicting the output than the others, they are weighted more while forming consensus opinions.

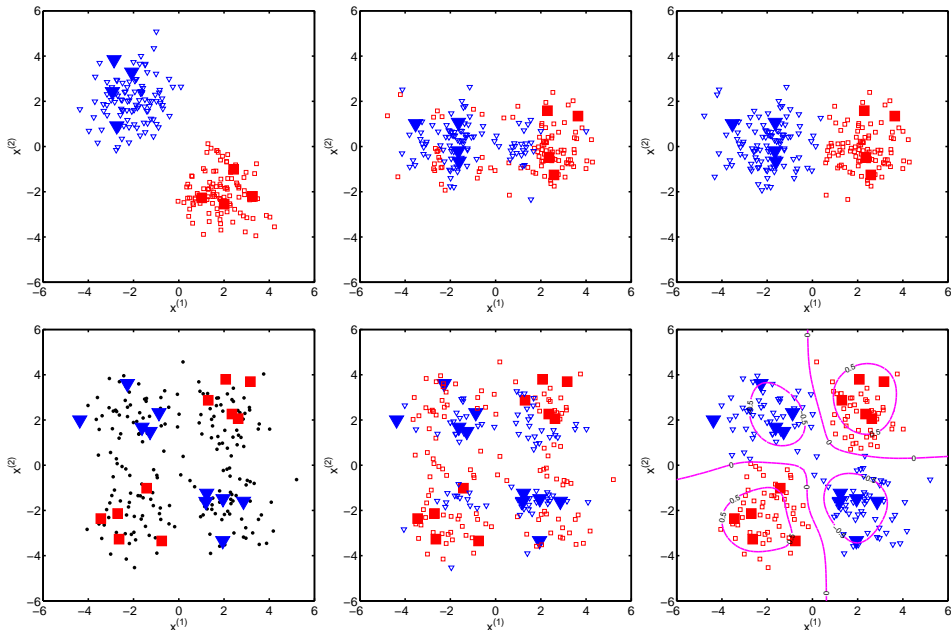

Figure 2: Toy examples for co-training. Big red/blue markers denote $+1/-1$ labeled points; remaining points are unlabeled. TOP **left**: co-training result on two-Gaussian data with mean $(2, -2)$ and $(-2, 2)$; **center** and **right**: canonical and Bayesian co-training on two-Gaussian data with mean $(2, 0)$ and $(-2, 0)$; BOTTOM **left**: XOR data with four Gaussians; **center** and **right**: Bayesian co-training and pure GP supervised learning result (with RBF kernel). Co-training is much worse than GP supervised learning in this case. All Gaussians have unit variance. RBF kernel uses width 1 for supervised learning and $1/\sqrt{2}$ for each feature in two-view learning.

These uncertainties can be easily optimized in the GP framework by maximizing the marginal of output $\mathbf{y}$ (omitted in this paper due to space limit).

**Unsupervised and semi-supervised multi-view learning:** The proposed graphical model also motivates new methods for unsupervised multi-view learning such as spectral clustering. While the similarity matrix of each view $j$ is encoded in $\mathbf{K}_j$, the co-training kernel $\mathbf{K}_c$ encodes the similarity of two data samples *with multiple views*, and thus can be used directly in spectral clustering. The extension to semi-supervised learning is also straightforward since $\mathbf{K}_c$ by definition depends on unlabeled data as well.

**Alternative interaction potential functions:** Previous discussions about multi-view learning rely on potential definitions in (3) (which we call the *consensus-based potentials*), but other definitions are also possible and will lead to different co-training models. Actually, the definition in (3) has fundamental limitations and leads only to consensus-based learning, as seen from the next subsection.

### 3.6 Limitations of Consensus-based Potentials

As mentioned before, the consensus-based potentials in (3) can be interpreted as defining a Gaussian prior (4) to $\mathbf{f}_c$, where the mean is a *weighted average* of the $m$ individual views. This averaging indicates that the value of $\mathbf{f}_c$ is never higher (or lower) than that of any single view. While the consensus-based potentials are intuitive and useful for many applications, they are limited for some real world problems where the evidence from different views should be *additive* (or enhanced) rather than averaging. For instance, when a radiologist is making a diagnostic decision about a lung cancer patient, he might look at both the CT image and the MRI image. If either of the two images gives a strong evidence of cancer, he can make decision based on a single view; if both images give an evidence of 0.6 (in a [0,1] scale), the final evidence of cancer should be higher (say, 0.8) than either of them. It's clear that the multi-view learning in this scenario is not consensus-based. While all the previously proposed co-training and co-regularization algorithms have thus far been based on enforcing consensus between the views, in principle our graphical model allows other forms of

Table 1: Results for Citeseer with different numbers of training data (pos/neg). Bold face indicates best performance. Bayesian co-training is significantly better than the others (p-value 0.01 in Wilcoxon rank sum test) except in AUC with "Train +2/-10".

| | # TRAIN +2/-10 | | # TRAIN +4/-20 | |
|---|---|---|---|---|
| MODEL | AUC | F1 | AUC | F1 |
| TEXT | $0.5725 \pm 0.0180$ | $0.1359 \pm 0.0565$ | $0.5770 \pm 0.0209$ | $0.1443 \pm 0.0705$ |
| INBOUND LINK | $0.5451 \pm 0.0025$ | $0.3510 \pm 0.0011$ | $0.5479 \pm 0.0035$ | $0.3521 \pm 0.0017$ |
| OUTBOUND LINK | $0.5550 \pm 0.0119$ | $0.3552 \pm 0.0053$ | $0.5662 \pm 0.0124$ | $0.3600 \pm 0.0059$ |
| TEXT+LINK | $0.5730 \pm 0.0177$ | $0.1386 \pm 0.0561$ | $0.5782 \pm 0.0218$ | $0.1474 \pm 0.0721$ |
| CO-TRAINED GPLR | $0.6459 \pm 0.1034$ | $0.4001 \pm 0.2186$ | $0.6519 \pm 0.1091$ | $0.4042 \pm 0.2321$ |
| BAYESIAN CO-TRAINING | $\mathbf{0.6536 \pm 0.0419}$ | $\mathbf{0.4210 \pm 0.0401}$ | $\mathbf{0.6880 \pm 0.0300}$ | $\mathbf{0.4530 \pm 0.0293}$ |

relationships between the views. In particular, potentials other than those in (3) should be of great interest for future research.

## 4 Experimental Study

**Toy Examples:** We show some 2D toy classification problems to visualize the co-training result (in Fig. 2). Our first example is a two-Gaussian case where either feature $x^{(1)}$ or $x^{(2)}$ can fully solve the problem (top left). This is an ideal case for co-training since: 1) each single view is sufficient to train a classifier, and 2) both views are conditionally independent given the class labels. The second toy data is a bit harder since the two Gaussians are aligned to the $x^{(1)}$-axis. In this case the feature $x^{(2)}$ is totally irrelevant to the classification problem. The canonical co-training fails here (top center) since when we add labels using the $x^{(2)}$ feature , noisy labels will be introduced and expanded to future training. The proposed model can handle this situation since we can adapt the weight of each view and penalize the feature $x^{(2)}$ (top right). Our third toy data follows an XOR shape where four Gaussians form a binary classification problem that is not linearly separable (bottom left). In this case both assumptions mentioned above are violated, and co-training failed completely (bottom center). A supervised learning model can however easily recover the non-linear underlying structure (bottom right). This indicates that the co-training kernel $\mathbf{K}_c$ is not suitable for this problem.

**Web Data:** We use two sets of linked documents for our experiment. The *Citeseer* data set contains 3,312 entries that belong to six classes. There are three natural views: the text view consists of title and abstract of a paper; the two link views are inbound and outbound references. We pick up the largest class which contains 701 documents and test the one-vs-rest classification performance. The *WebKB* data set is a collection of 4,502 academic web pages manually grouped into six classes (student, faculty, staff, department, course, project). There are two views containing the text on the page and the anchor text of all inbound links, respectively. We consider the binary classification problem "student" against "faculty", for which there are 1,641 and 1,119 documents, respectively.

We compare the single-view learning methods (TEXT, INBOUND LINK, etc), concatenated-view method (TEXT+LINK), and co-training methods CO-TRAINED GPLR (Co-Trained Gaussian Process Logistic Regression) and BAYESIAN CO-TRAINING. Linear kernels are used for all the competing methods. For the canonical co-training method we repeat 50 times and in each iteration add the most predictable 1 positive sample and $r$ negative samples into the training set where $r$ depends on the number of negative/positive ratio of each data set. Performance is evaluated using AUC score and F1 measure. We vary the number of training documents (with ratio proportional to the true positive/negative ratio), and all the co-training algorithms use all the unlabeled data in the training process. The experiments are repeated 20 times and the prediction means and standard deviations are shown in Table 1 and 2.

It can be seen that for Citeseer the co-training methods are better than the supervised methods. In this cases Bayesian co-training is better than canonical co-training and achieves the best performance. For WebDB, however, canonical co-trained GPLR is not as good as supervised algorithms, and thus Bayesian co-training is also worse than supervised methods though a little better than co-trained GPLR. This is maybe because the TEXT and LINK features are not independent given the class labels (especially when two classes "faculty" and "staff" might share features). Canonical co-training has higher deviations than other methods due to the possibility of adding noisy labels. We have also tried other number of iterations but 50 seems to give an overall best performance.

Table 2: Results for WebKB with different numbers of training data (pos/neg). Bold face indicates best performance. No results are significantly better than all the others (p-value 0.01 in Wilcoxon rank sum test).

| MODEL | # TRAIN +2/-2 | | # TRAIN +4/-4 | |
|---|---|---|---|---|
| | AUC | F1 | AUC | F1 |
| TEXT | $0.5767 \pm 0.0430$ | $0.4449 \pm 0.1614$ | $\mathbf{0.6150 \pm 0.0594}$ | $0.5338 \pm 0.1267$ |
| INBOUND LINK | $0.5211 \pm 0.0017$ | $\mathbf{0.5761 \pm 0.0013}$ | $0.5210 \pm 0.0019$ | $\mathbf{0.5758 \pm 0.0015}$ |
| TEXT+LINK | $0.5766 \pm 0.0429$ | $0.4443 \pm 0.1610$ | $\mathbf{0.6150 \pm 0.0594}$ | $0.5336 \pm 0.1267$ |
| CO-TRAINED GPLR | $0.5624 \pm 0.1058$ | $0.5437 \pm 0.1225$ | $0.5959 \pm 0.0927$ | $0.5737 \pm 0.1203$ |
| BAYESIAN CO-TRAINING | $\mathbf{0.5794 \pm 0.0491}$ | $0.5562 \pm 0.1598$ | $0.6140 \pm 0.0675$ | $0.5742 \pm 0.1298$ |

Note that the single-view learning with TEXT almost achieves the same performance as concatenated-view method. This is because the number of text features are much more than the link features (e.g., for WebKB there are 24,480 text features and only 901 link features). So these multiple views are very unbalanced and should be taken into account in co-training with different weights. Bayesian co-training provides a natural way of doing it.

## 5 Conclusions

This paper has two principal contributions. We have proposed a graphical model for combining multi-view data, and shown that previously derived co-regularization based training algorithms maximize the likelihood of this model. In the process, we showed that these algorithms have been making an intrinsic assumption of the form $p(f_c, f_1, f_2, \ldots, f_m) \propto \psi(f_c, f_1)\psi(f_c, f_2) \ldots \psi(f_c, f_m)$, even though it was not explicitly realized earlier. We also studied circumstances when this assumption proves unreasonable. Thus, our first contribution was to clarify the implicit assumptions and limitations in multi-view consensus learning in general, and co-regularization in particular.

Motivated by the insights from the graphical model, our second contribution was the development of alternative algorithms for co-regularization; in particular the development of a non-stationary co-training kernel, and the development of methods for using side-information in classification. Unlike previously published co-regularization algorithms, our approach: (a) handles naturally more than 2 views; (b) automatically learns which views of the data should be trusted more while predicting class labels; (c) shows how to leverages previously developed methods for efficiently training GP/SVM; (d) clearly explains our assumptions, what is being optimized *overall*, etc; (e) does not suffer from local maxima problems; (f) is less computationally demanding in terms of both speed and memory requirements.

## Footnotes

[1]The definition of $\psi$ in this paper has been overloaded to simplify notation, but its meaning should be clear from the function arguments.

## References

[1] S. Dasgupta, M. Littman, and D. McAllester. PAC generalization bounds for co-training. In *NIPS*, 2001.

[2] A. Blum and T. Mitchell. Combining labeled and unlabeled data with co-training. In *COLT*, 1998.

[3] N. Balcan, A. Blum, and K. Yang. Co-training and expansion: Towards bridging theory and practice. In *NIPS*, 2004.

[4] K. Nigam and R. Ghani. Analyzing the effectiveness and applicability of co-training. In *Workshop on information and knowledge management*, 2000.

[5] U. Brefeld and T. Scheffer. Co-em support vector learning. In *ICML*, 2004.

[6] Steffen Bickel and Tobias Scheffer. Estimation of mixture models using co-em. In *ECML*, 2005.

[7] B. Krishnapuram, D. Williams, Y. Xue, A. Hartemink, L. Carin, and M. Figueiredo. On semi-supervised classification. In *NIPS*, 2004.

[8] Virginia de Sa. Spectral clustering with two views. In *ICML Workshop on Learning With Multiple Views*, 2005.

[9] C. E. Rasmussen and C. K. I. Williams. *Gaussian Processes for Machine Learning*. MIT Press, 2006.

[10] Xiaojin Zhu, John Lafferty, and Zoubin Ghahramani. Semi-supervised learning: From Gaussian fields to gaussian processes. Technical report, CMU-CS-03-175, 2003.

